# Cross-Validation Optimization for Large Scale Hierarchical Classification Kernel Methods

**Matthias W. Seeger**
Max Planck Institute for Biological Cybernetics
P.O. Box 2169, 72012 Tübingen, Germany
seeger@tuebingen.mpg.de

## Abstract

We propose a highly efficient framework for kernel multi-class models with a large and structured set of classes. Kernel parameters are learned automatically by maximizing the cross-validation log likelihood, and predictive probabilities are estimated. We demonstrate our approach on large scale text classification tasks with hierarchical class structure, achieving state-of-the-art results in an order of magnitude less time than previous work.

## 1   Introduction

In many real-world statistical problems, we would like to fit a model with a large number of dependent variables to a training sample with very many cases. For example, in multi-way classification problems with a structured label space, modern applications demand predictions on thousands of classes, and very large datasets become available. If $n$ and $C$ denote dataset size and number of classes respectively, nonparametric kernel methods like SVMs or Gaussian processes typically scale superlinearly in $n\,C$, if dependencies between the latent class functions are properly represented. Furthermore, most large scale kernel methods proposed so far refrain from solving the problem of learning hyperparameters (kernel or loss function parameters). The user has to run cross-validation schemes, which require frequent human interaction and are not suitable for learning more than a few hyperparameters.

In this paper, we propose a general framework for learning in probabilistic kernel classification models. While the basic model is standard, a major feature of our approach is the high computational efficiency with which the primary fitting (for fixed hyperparameters) is done, allowing us to deal with hundreds of classes and thousands of datapoints within a few minutes. The primary fitting scales linearly in $C$, and depends on $n$ mainly via a fixed number of *matrix-vector multiplications* (MVM) with $n \times n$ kernel matrices. In many situations, these MVM primitives can be computed very efficiently, as will be demonstrated. Furthermore, we optimize hyperparameters *automatically* by minimizing the cross-validation log likelihood, making use of our primary fitting technology as inner loop in order to compute the CV criterion and its gradient. Our approach can be used to learn a large number of hyperparameters and does not need user interaction.

Our framework is generally applicable to structured label spaces, which we demonstrate here for *hierarchical classification* of text documents. The hierarchy is represented through an ANOVA setup. While the $C$ latent class functions are fully dependent *a priori*, the scaling of our method stays within a factor of two compared to unstructured classification. We test our framework on the same tasks treated in [1], achieving comparable results in at least an order of magnitude less time. Our method estimates predictive probabilities for each test point, which can allow better predictions w.r.t. loss functions different from zero-one.

The primary fitting method is given in Section 2, the extension to hierarchical classification in Section 3. Hyperparameter learning is discussed in Section 4. Computational details are provided in

Section 5. We present experimental results in Section 6. Our highly efficient implementation is publicly available, as project *klr* in the *LHOTSE*[1] toolbox for adaptive statistical models.

## 2  Penalized Multiple Logistic Regression

Our problem is to predict $y \in \{1, \ldots, C\}$ from $\boldsymbol{x} \in \mathcal{X}$, given some i.i.d. data $D = \{(\boldsymbol{x}_i, \boldsymbol{y}_i) \,|\, i = 1, \ldots, n\}$. We use zero-one coding, *i.e.* $\boldsymbol{y}_i \in \{0,1\}^C$, $\mathbf{1}^T \boldsymbol{y}_i = 1$. We elpoy the *multiple logistic regression model*, consisting of $C$ latent (unobserved) class functions $u_c$ feeding into the multiple logistic (or softmax) likelihood $P(y_{i,c} = 1 | \boldsymbol{x}_i, \boldsymbol{u}_i) = e^{u_c(\boldsymbol{x}_i)} / (\sum_{c'} e^{u_{c'}(\boldsymbol{x}_i)})$. We write $u_c = f_c + b_c$ for intercept parameters $b_c \in \mathbb{R}$ and functions $f_c$ living in a reproducing kernel Hilbert space (RKHS) with kernel $K^{(c)}$, and consider the *penalized negative log likelihood* $\Phi = -\sum_{i=1}^n \log P(\boldsymbol{y}_i | \boldsymbol{u}_i) + (1/2) \sum_{c=1}^C \|f_c\|_c^2 + (1/2)\sigma^{-2} \|\boldsymbol{b}\|^2$, which we minimize for primary fitting. $\|\cdot\|_c$ is the RKHS norm for kernel $K^{(c)}$. Details on such setups can be found in [4].

Our notation for $n\,C$ vectors[2] (and matrices) uses the ordering $\boldsymbol{y} = (y_{1,1}, y_{2,1}, \ldots, y_{n,1}, y_{1,2}, \ldots)$. We set $\boldsymbol{u} = (u_c(\boldsymbol{x}_i)) \in \mathbb{R}^{nC}$. $\otimes$ denotes the Kronecker product, $\mathbf{1}$ is the vector of all ones. Selection indexes $I$ are applied to $i$ only: $\boldsymbol{y}_I = (y_{i,c})_{i \in I, c} \in \mathbb{R}^{|I|C}$.

Since the likelihood depends on the $f_c$ only through $f_c(\boldsymbol{x}_i)$, every minimizer of $\Phi$ must be a kernel expansion: $f_c = \sum_i \alpha_{i,c} K^{(c)}(\cdot, \boldsymbol{x}_i)$ (representer theorem, see [4]). Plugging this in, the regularizer becomes $(1/2)\boldsymbol{\alpha}^T \boldsymbol{K} \boldsymbol{\alpha} + (1/2)\sigma^{-2} \|\boldsymbol{b}\|^2$. $\boldsymbol{K}^{(c)} = (K^{(c)}(\boldsymbol{x}_i, \boldsymbol{x}_j))_{i,j} \in \mathbb{R}^{n,n}$, $\boldsymbol{K} = \mathrm{diag}(\boldsymbol{K}^{(c)})_c$ is block-diagonal. We refer to this setup as *flat classification* model. The $b_c$ may be eliminated as $\boldsymbol{b} = \sigma^2 (\boldsymbol{I} \otimes \mathbf{1}^T) \boldsymbol{\alpha}$. Thus, if $\tilde{\boldsymbol{K}} = \boldsymbol{K} + \sigma^2 (\boldsymbol{I} \otimes \mathbf{1})(\boldsymbol{I} \otimes \mathbf{1}^T)$, then $\Phi$ becomes

$$\Phi = \Phi_{lh} + \frac{1}{2}\boldsymbol{\alpha}^T \tilde{\boldsymbol{K}} \boldsymbol{\alpha}, \quad \Phi_{lh} = -\boldsymbol{y}^T \boldsymbol{u} + \mathbf{1}^T \boldsymbol{l}, \quad l_i = \log \mathbf{1}^T \exp(\boldsymbol{u}_i), \quad \boldsymbol{u} = \tilde{\boldsymbol{K}} \boldsymbol{\alpha}. \quad (1)$$

$\Phi$ is strictly convex in $\boldsymbol{\alpha}$ (because the likelihood is log-concave), so it has a unique minimum point $\hat{\boldsymbol{\alpha}}$. The corresponding kernel expansions are $\hat{u}_c = \sum_i \hat{\alpha}_{i,c}(K^{(c)}(\cdot, \boldsymbol{x}_i) + \sigma^2)$. Estimates of the conditional probability on test points $\boldsymbol{x}_*$ are obtained by plugging $\hat{u}_c(\boldsymbol{x}_*)$ into the likelihood.

We note that this setup can also be seen as MAP approximation to a Bayesian model, where the $f_c$ are given independent Gaussian process priors, *e.g.*[7]. It is also related to the multi-class SVM [2], where $-\log P(y_i | \boldsymbol{u}_i)$ is replaced by the margin loss $-u_{y_i}(\boldsymbol{x}_i) + \max_c \{u_c(\boldsymbol{x}_i) + 1 - \delta_{c,y_i}\}$. The negative log multiple logistic likelihood has similar properties, but is smooth as a function of $\boldsymbol{u}$, and the primary fitting of $\boldsymbol{\alpha}$ does not require constrained convex optimization.

We minimize $\Phi$ using the *Newton-Raphson* (NR) algorithm, the details are provided in Section 5. The complexity of our fitting algorithm is dominated by $k_1(k_2 + 2)$ matrix-vector multiplications with $\boldsymbol{K}$, where $k_1$ is the number of NR iterations, $k_2$ the number of *linear conjugate gradient* (LCG) steps for computing each Newton direction. Since NR is a second-order convergent method, $k_1$ can be chosen small. $k_2$ determines the quality of each Newton direction, for both fairly small values are sufficient (see Section 6.2).

## 3  Hierarchical Classification

So far we dealt with flat classification, the classes being independent *a priori*, with block-diagonal kernel matrix $\boldsymbol{K}$. However, if the label set has a known structure[3], we can benefit from representing it in the model. Here we focus on *hierarchical classification*, the label set $\{1, \ldots, C\}$ being the leaf nodes of a tree. Classes with lower common ancestor should be more closely related. In this Section, we propose a model for this setup and show how it can be dealt with in our framework with minor modifications and minor extra cost.

In flat classification, the latent class functions $u_c$ are modelled as *a priori* independent, in that the regularizer (which plays the role of a log prior) is a sum of individual terms for each $u_c$, without any

interaction terms. *Analysis of variance* (ANOVA) models go beyond this independent design, they have previously been applied to text classification by [1]. Let $\{0, \ldots, P\}$ be the nodes of the tree, $0$ being the root, and the numbers are assigned breadth first $(1, 2, \ldots$ are the root's children). The tree is determined by $P$ and $n_p$, $p = 0, \ldots, P$, the number of children of node $p$. Let $L$ be the set of leaf nodes, $|L| = C$. Assign a *pair* of latent functions $u_p$, $\breve{u}_p$ to each node, except the root. The $\breve{u}_p$ are assumed *a priori* independent, as in flat classification. $u_p$ is the sum of $\breve{u}_{p'}$, $p'$ running over the nodes (including $p$) on the path from the root to $p$. The class functions to be fed into the likelihood are the $u_{L(c)}$ of the leafs. This setup represents similarities conditioned on the hierarchy. For example, if leafs $L(c)$, $L(c')$ have the common parent $p$, then $u_{L(c)} = u_p + \breve{u}_{L(c)}$, $u_{L(c')} = u_p + \breve{u}_{L(c')}$, so the class functions *share* the effect $u_p$. Since regularization forces all independent effects $\breve{u}_{p'}$ to be smooth, the classes $c$, $c'$ are urged to behave similarly *a priori*.

Let $\boldsymbol{u} = (u_p(\boldsymbol{x}_i))_{i,p}$, $\breve{\boldsymbol{u}} = (\breve{u}_p(\boldsymbol{x}_i))_{i,p} \in \mathbb{R}^{nP}$. The vectors are related as $\boldsymbol{u} = (\boldsymbol{\Phi} \otimes \boldsymbol{I})\breve{\boldsymbol{u}}$, $\boldsymbol{\Phi} \in \{0,1\}^{P,P}$. Importantly, $\boldsymbol{\Phi}$ has a simple structure which allows MVM with $\boldsymbol{\Phi}$ or $\boldsymbol{\Phi}^T$ to be computed easily in $O(P)$, without having to compute or store $\boldsymbol{\Phi}$ explicitly. MVM with $\boldsymbol{\Phi}$ is described in Algorithm 1, and MVM with $\boldsymbol{\Phi}^T$ works in a similar manner [8].

Under the hierarchical model, the class functions $u_{L(c)}$ are strongly dependent *a priori*. We may represent this prior coupling in our framework by simply plugging in the implied kernel matrix $\boldsymbol{K}$:

$$\boldsymbol{K} = (\boldsymbol{\Phi}_{L,\cdot} \otimes \boldsymbol{I})\breve{\boldsymbol{K}}(\boldsymbol{\Phi}_{L,\cdot}^T \otimes \boldsymbol{I}), \tag{2}$$

where the inner $\breve{\boldsymbol{K}}$ is block-diagonal. $\boldsymbol{K}$ is not sparse and certainly not block-diagonal, but the important point is that we are still able to do kernel MVMs efficiently: pre- and postmultiplying by $\boldsymbol{\Phi}$ is cheap, and $\breve{\boldsymbol{K}}$ is block-diagonal just as in the flat case.

We note that the step from flat to hierarchical classification requires minor modifications of existing code only. If code for representing a block-diagonal $\boldsymbol{K}$ is available, we can use it to represent the inner $\breve{\boldsymbol{K}}$, just replacing $C$ by $P$. This simplicity carries through to the hyperparameter learning case (see Section 4). The cost of a kernel MVM is increased by a factor $P/C < 2$, which in most hierarchies in practice is close to 1. However, it would be wrong to claim that hierarchical classification in general comes as cheap as flat classification.

---

**Algorithm 1: Matrix-vector multiplication**
$\boldsymbol{y} = \boldsymbol{\Phi}\boldsymbol{x}$

$\boldsymbol{y} \leftarrow ().\, y_0 := 0.\, s := 0.$
**for** $p = 0, \ldots, P$ **do**
  **if** $n_p > 0$ ($p$ not a leaf node) **then**
    Let $J(p) = \{s+1, \ldots, s+n_p\}$.
    $\boldsymbol{y} \leftarrow (\boldsymbol{y}^T,\ y_p\boldsymbol{1}^T + \boldsymbol{x}_{J(p)}^T)^T.\, s \leftarrow s + n_p.$
  **end if**
**end for**

---

The subtle issue is that the primary fitting becomes more costly, precisely because there is more coupling between the variables. In the flat case, the Hessian of $\Phi$ is close to block-diagonal. The LCG algorithm to compute Newton directions converges quickly, because it nearly decomposes into $C$ independent ones, and fewer NR steps are required (see Section 5). In the hierarchical case, both LCG and NR need more iterations to attain the same accuracy. In numerical mathematics, much work has been done to approximately decouple linear systems by *preconditioning*. In some of these strategies, knowledge about the structure of the system matrix (in our case: the hierarchy) can be used to drive preconditioning. An important point for future research is to find a good preconditioning strategy for the system of Eq. 5. However, in all our experiments so far the fitting of the hierarchical model took less than twice the time required for the flat model on the same task. Some further extensions, such as learning with incomplete label information, are discussed in [8].

## 4 Hyperparameter Learning

In any model of interest, there will be free *hyperparameters* $\boldsymbol{h}$, for example parameters of the kernels $K^{(c)}$. These were assumed to be fixed in the primary fitting method introduced in Section 2. In this Section, we describe a scheme for learning $\boldsymbol{h}$ which makes use of the primary fitting algorithm as inner loop. Note that such nested strategies are commonplace in Bayesian Statistics, where (marginal) inference is typically used as subroutine for parameter learning.

Recall that primary fitting consists of minimizing $\Phi$ of Eq. 1 w.r.t. $\boldsymbol{\alpha}$. If we minimize $\Phi$ w.r.t. $\boldsymbol{h}$ as well, we run into the problem of overfitting. A common remedy is to minimize the negative *cross-*

*validation log likelihood* $\Psi$ instead. Let $\{I_k\}$ be a partition of $\{1, \ldots, n\}$, with $J_k = \{1, \ldots, n\} \setminus I_k$, and let $\Phi_{J_k} = \boldsymbol{u}_{[J_k]}^T ((1/2)\boldsymbol{\alpha}_{[J_k]} - \boldsymbol{y}_{J_k}) + \mathbf{1}^T \boldsymbol{l}_{[J_k]}$ be the primary criterion on the subset $J_k$ of the data. Here, $\boldsymbol{u}_{[J_k]} = \tilde{\boldsymbol{K}}_{J_k} \boldsymbol{\alpha}_{[J_k]}$. The $\boldsymbol{\alpha}_{[J_k]}$ are independent variables, *not* part of a common $\boldsymbol{\alpha}$. The CV criterion is

$$\Psi = \sum_k \Psi_{I_k}, \quad \Psi_{I_k} = -\boldsymbol{y}_{I_k}^T \boldsymbol{u}_{[I_k]} + \mathbf{1}^T \boldsymbol{l}_{[I_k]}, \quad \boldsymbol{u}_{[I_k]} = \tilde{\boldsymbol{K}}_{I_k, J_k} \boldsymbol{\alpha}_{[J_k]}, \tag{3}$$

where $\boldsymbol{\alpha}_{[J_k]}$ minimizes $\Phi_{J_k}$. Since for each $k$, we fit and evaluate on disjoint parts of $\boldsymbol{y}$, $\Psi$ is an unbiased estimator of the test negative log likelihood, and minimizing $\Psi$ should be robust to overfitting.

In order to select $\boldsymbol{h}$, we pick a fixed partition at random, then do gradient-based minimization of $\Psi$ w.r.t. $\boldsymbol{h}$. To this end, we keep the set $\{\boldsymbol{\alpha}_{[J_k]}\}$ of primary variables, and iterate between re-fitting those for each fold $I_k$, and computing $\Psi$ and $\nabla_{\boldsymbol{h}} \Psi$. The latter can be determined analytically, requiring us to solve a linear system with the Hessian matrix $\boldsymbol{I} + \boldsymbol{V}_{[J_k]}^T \tilde{\boldsymbol{K}}_{J_k} \boldsymbol{V}_{[J_k]}$ already encountered during primary fitting (see Section 5). This means that the same LCG code used to compute Newton directions there can be applied here in order to compute the gradient of $\Psi$. The details are given in Section 5. As for the complexity, suppose there are $q$ folds. The update of the $\boldsymbol{\alpha}_{[J_k]}$ requires $q$ primary fitting applications, but since they are initialized with the previous values $\boldsymbol{\alpha}_{[J_k]}$, they do converge very rapidly, especially during later outer iterations. Computing $\Psi$ based on the $\boldsymbol{\alpha}_{[J_k]}$ comes basically for free. The gradient computation decomposes into two parts: accumulation, and kernel derivative MVMs. The accumulation part requires solving $q$ systems of size $((q-1)/q)n\,C$, thus $q\,k_3$ kernel MVMs on the $\tilde{\boldsymbol{K}}_{J_k}$ if linear conjugate gradients (LCG) is used, $k_3$ being the number of LCG steps. We also need two buffer matrices $\boldsymbol{E}, \boldsymbol{F}$ of $q\,n\,C$ elements each. Note that the accumulation step is *independent* of the number of hyperparameters. The kernel derivative MVM part consists of $q$ derivative MVM calls for each independent component of $\boldsymbol{h}$, see Section 5.1. As opposed to the accumulation part, this part consists of a simple large matrix operation and can be run very efficiently using specialized numerical linear algebra code.

As shown in Section 5, the extension of hyperparameter learning to the hierarchical case of Section 3 is simply done by wrapping the accumulation part, the coding and additional memory effort being minimal. Given a method for computing $\Psi$ and $\nabla_{\boldsymbol{h}} \Psi$, we plug these into a custom optimizer such as Quasi-Newton in order to learn $\boldsymbol{h}$.

## 5 Computational Details

In this Section, we provide details for the general plan laid out above. It is precisely these which characterize our framework and allow us to apply a standard model to domains beyond its usual applications, but of interest to Machine Learning.

Recall Section 2. We minimize $\Phi$ by choosing search directions $\boldsymbol{s}$, and doing line minimizations along $\boldsymbol{\alpha} + \lambda \boldsymbol{s}, \ \lambda > 0$. For the latter, we maintain the pair $(\boldsymbol{\alpha}, \boldsymbol{u}), \ \boldsymbol{u} = \tilde{\boldsymbol{K}} \boldsymbol{\alpha}$. We have:

$$\nabla_{\boldsymbol{u}} \Phi = \boldsymbol{\pi} - \boldsymbol{y} + \boldsymbol{\alpha}, \quad \boldsymbol{\pi} = \exp(\boldsymbol{u} - \mathbf{1} \otimes \boldsymbol{l}), \ i.e. \ \pi_{i,c} = P(y_{i,c} = 1|\boldsymbol{u}_i). \tag{4}$$

Given $(\boldsymbol{\alpha}, \boldsymbol{u})$, $\Phi$ and $\nabla_{\boldsymbol{u}} \Phi$ can be computed in $O(n\,C)$, without requiring MVMs. This suggests to perform the line search in $\boldsymbol{u}$ along the direction $\tilde{\boldsymbol{s}} = \tilde{\boldsymbol{K}} \boldsymbol{s}$, the corresponding $\boldsymbol{\alpha}$ can be constructed from the final $\lambda$. Since kernel MVMs are significantly more expensive than these $O(n\,C)$ operations, the line searches basically come for free!

We choose search directions by *Newton-Raphson* (NR)[4], since the Hessian of $\Phi$ is required anyway for hyperparameter learning. Let $\boldsymbol{D} = \operatorname{diag} \boldsymbol{\pi}, \ \boldsymbol{P} = (\mathbf{1} \otimes \boldsymbol{I})(\mathbf{1}^T \otimes \boldsymbol{I})$, and $\boldsymbol{W} = \boldsymbol{D} - \boldsymbol{D} \boldsymbol{P} \boldsymbol{D}$. We have $\nabla\nabla_{\boldsymbol{u}} \Phi_{lh} = \boldsymbol{W}$, and $\boldsymbol{g} = \nabla_{\boldsymbol{u}} \Phi_{lh} = \boldsymbol{\pi} - \boldsymbol{y}$ from Eq. 4. The NR system is $(\boldsymbol{I} + \boldsymbol{W} \tilde{\boldsymbol{K}}) \boldsymbol{\alpha}' = \boldsymbol{W} \boldsymbol{u} - \boldsymbol{g}$, with the NR direction being $\boldsymbol{s} = \boldsymbol{\alpha}' - \boldsymbol{\alpha}$. If $\boldsymbol{V} = (\boldsymbol{I} - \boldsymbol{D} \boldsymbol{P}) \boldsymbol{D}^{1/2}$, then $\boldsymbol{W} = \boldsymbol{V} \boldsymbol{V}^T$, because $(\mathbf{1}^T \otimes \boldsymbol{I}) \boldsymbol{D} = \boldsymbol{I}$. We see that $\boldsymbol{\alpha}' = \boldsymbol{V} \boldsymbol{\beta}$ (using $(\mathbf{1}^T \otimes \boldsymbol{I}) \boldsymbol{g} = \boldsymbol{0}$), and we can obtain it from the equivalent *symmetric system*

$$\left(\boldsymbol{I} + \boldsymbol{V}^T \tilde{\boldsymbol{K}} \boldsymbol{V}\right) \boldsymbol{\beta} = \boldsymbol{V}^T \boldsymbol{u} - \boldsymbol{D}^{-1/2}(\boldsymbol{\pi} - \boldsymbol{y}), \quad \boldsymbol{\alpha}' = \boldsymbol{V} \boldsymbol{\beta} \tag{5}$$

(details are in [8]). Note that $\boldsymbol{Px} = (\sum_{c'} \boldsymbol{x}^{(c')})_c$, so that MVM with $\boldsymbol{V}$ can be done in $O(n\,C)$. The NR direction is obtained by solving this system approximately by the *linear conjugate gradients* (LCG) method, requiring a MVM with the system matrix in each iteration, thus a single MVM with $\boldsymbol{K}$. Our implementation includes diagonal preconditioning and numerical stability safeguards [8]. The NR system need not be solved to high accuracy (see Section 6.2). Initially, $\boldsymbol{\beta} = \boldsymbol{D}^{-1/2}\boldsymbol{\alpha}$, because then $\boldsymbol{V}\boldsymbol{\beta} = \boldsymbol{\alpha}$ if only $(\boldsymbol{1}^T \otimes \boldsymbol{I})\boldsymbol{\alpha} = \boldsymbol{0}$, which is true if the initial $\boldsymbol{\alpha}$ fulfils it.

We now show how to compute the gradient $\nabla_{\boldsymbol{h}}\Psi$ for the CV criterion $\Psi$ (Eq. 3). Note that $\boldsymbol{\alpha}_{[J]}$ is determined by the stationary equation $\boldsymbol{\alpha}_{[J]} + \boldsymbol{g}_{[J]} = \boldsymbol{0}$. Taking the derivative gives $d\boldsymbol{\alpha}_{[J]} = -\boldsymbol{W}_{[J]}((d\boldsymbol{K}_J)\boldsymbol{\alpha}_{[J]} + \tilde{\boldsymbol{K}}_J(d\boldsymbol{\alpha}_{[J]}))$. We obtain a system for $d\boldsymbol{\alpha}_{[J]}$ which is symmetrized as above: $(\boldsymbol{I} + \boldsymbol{V}_{[J]}^T \tilde{\boldsymbol{K}}_J \boldsymbol{V}_{[J]})\boldsymbol{\beta} = -\boldsymbol{V}_{[J]}^T(d\boldsymbol{K}_J)\boldsymbol{\alpha}_{[J]}$, $d\boldsymbol{\alpha}_{[J]} = \boldsymbol{V}_{[J]}\boldsymbol{\beta}$. Also, $d\Psi_I = (\boldsymbol{\pi}_{[I]} - \boldsymbol{y}_I)^T((d\boldsymbol{K}_{I,J})\boldsymbol{\alpha}_{[J]} + \tilde{\boldsymbol{K}}_{I,J}(d\boldsymbol{\alpha}_{[J]}))$. With $\boldsymbol{s} = \boldsymbol{I}_{\cdot,I}(\boldsymbol{\pi}_{[I]} - \boldsymbol{y}_I) - \boldsymbol{I}_{\cdot,J}\boldsymbol{V}_{[J]}(\boldsymbol{I} + \boldsymbol{V}_{[J]}^T \tilde{\boldsymbol{K}}_J \boldsymbol{V}_{[J]})^{-1}\boldsymbol{V}_{[J]}^T \tilde{\boldsymbol{K}}_{J,I}(\boldsymbol{\pi}_{[I]} - \boldsymbol{y}_I)$, we have that $d\Psi_I = (\boldsymbol{I}_{\cdot,J}\boldsymbol{\alpha}_{[J]})^T(d\boldsymbol{K})\boldsymbol{s}$. If we collect these vectors as columns of $\boldsymbol{E}$, $\boldsymbol{F} \in \mathbb{R}^{nC,q}$, we have that $d\Psi = \operatorname{tr}\boldsymbol{E}^T(d\boldsymbol{K})\boldsymbol{F}$. In the hierarchical setup, we use Eq. 2: $\tilde{\boldsymbol{E}} = (\boldsymbol{\Phi}_{L,\cdot}^T \otimes \boldsymbol{I})\boldsymbol{E} \in \mathbb{R}^{nP,q}$, $\tilde{\boldsymbol{F}}$ accordingly, then $d\Psi = \operatorname{tr}\tilde{\boldsymbol{E}}^T(d\breve{\boldsymbol{K}})\tilde{\boldsymbol{F}}$. Here, we build $\boldsymbol{E}$, $\boldsymbol{F}$ in the buffers allocated for $\tilde{\boldsymbol{E}}$, $\tilde{\boldsymbol{F}}$, then transform them later in place.

We finally mention some of the computational "tricks", without which we could not have dealt with the largest tasks in Section 6.2 (for section B, a single $n\,C$ vector requires $88M$ of memory). For the linear kernel (see Section 5.1), the main primitive $\boldsymbol{A} \mapsto \boldsymbol{X}\boldsymbol{X}^T\boldsymbol{A}$ can be coded very efficiently using a standard sparse matrix format for $\boldsymbol{X}$. If $\boldsymbol{A}$ is stored row-major ($a_{1,1}, a_{1,2}, \dots$), the computation becomes faster by a factor of $4$ to $6$ compared to the standard column-major format[5]. For hyperparameter learning, we work on subsets $J_k$ and need MVMs with $\tilde{\boldsymbol{K}}_{J_k}$. "Covariance representation shuffling" permutes the representation s.t. $\tilde{\boldsymbol{K}}_{J_k}$ sits in the upper left part, and MVM can use flat rather than indexed code, which is many times faster. We also share memory blocks of size $n\,C$ between LCG, gradient accumulation, line searches, keeping the overall memory requirements at $r\,n\,C$ for a small constant $r$, and avoiding frequent reallocations.

## 5.1 Matrix-Vector Multiplication

MVM with $\boldsymbol{K}$ is the bottleneck of our framework, and all efforts should be concentrated on this primitive. We can tap into much prior work in numerical mathematics. With many classes $C$, we may share kernels: $K^{(c)} = v_c M^{(l_c)}$, $v_c > 0$ variance parameters, $M^{(l)}$ independent correlation functions. Our generic implementation stores two symmetric matrices $\boldsymbol{M}^{(l)}$ in a $n \times n$ buffer.

The *linear kernel* $K^{(c)}(\boldsymbol{x}, \boldsymbol{x}') = v_c \boldsymbol{x}^T \boldsymbol{x}'$ is frequently used for text classification (see Section 6.2). If the data matrix $\boldsymbol{X}$ is sparse, kernel MVM can be done in much less than the generic $O(C\,n^2)$, typically in $O(C\,n)$, requiring $O(n)$ storage for $\boldsymbol{X}$ only, even if the dimension of $\boldsymbol{x}$ is way beyond $n$.

If the $K^{(c)}$ are isotropic kernels (depending on $\|\boldsymbol{x} - \boldsymbol{x}'\|$ only) and the $\boldsymbol{x}$ are low-dimensional, MVM with $\boldsymbol{K}^{(c)}$ can be approximated using specialized nearest neighbour data structures such as *KD trees* [12, 9]. Again, the MVM cost is typically $O(C\,n)$ in this case. For general kernels whose kernel matrices have a rapidly decaying eigenspectrum, one can approximate MVM by using *low-rank matrices* instead of the $\boldsymbol{K}^{(c)}$ [10], whence MVM is $O(C\,n\,d)$, $d$ the rank.

In Section 4 we also need MVM with the derivatives $(\partial/\partial h_j)\boldsymbol{K}^{(c)}$. Note that $(\partial/\partial \log v_c)\boldsymbol{K}^{(c)} = \boldsymbol{K}^{(c)}$, reducing to kernel MVM. For isotropic kernels, $\boldsymbol{K}^{(c)} = f(\boldsymbol{A})$, $a_{i,j} = \|\boldsymbol{x}_i - \boldsymbol{x}_j\|$, so $(\partial/\partial h_j)\boldsymbol{K}^{(c)} = g_j(\boldsymbol{A})$. If KD trees are used to approximate $\boldsymbol{A}$, they can be used equivalently (and with little additional cost) for computing derivative MVMs.

# 6 Experiments

In this Section, we provide experimental results for our framework on data from remote sensing, and on a set of large text classification tasks with very many classes, the latter are hierarchical.

## 6.1 Flat Classification: Remote Sensing

We use the *satimage* remote sensing task from the *statlog* repository.[6] This task has been used in the extensive SVM multi-class study of [5], where it is among the datasets on which the different methods show the most variance. It has $n = 4435$ training, $m = 2000$ test cases, and $C = 6$ classes. We use the isotropic *Gaussian (RBF)* kernel

$$K^{(c)}(\boldsymbol{x}, \boldsymbol{x}') = v_c \exp\left(-\frac{w_c}{2d}\|\boldsymbol{x} - \boldsymbol{x}'\|^2\right), \quad v_c, w_c > 0, \quad \boldsymbol{x}, \boldsymbol{x}' \in \mathbb{R}^d. \tag{6}$$

We compare the methods *mc-sep* (ours with separate kernels for each class; 12 hyperparameters), *mc-tied* (ours with a single shared kernel; 2 hyperparameters), *1rest* (one-against-rest: $C$ binary classifiers are trained separately to discriminate $c$ from the rest, they are voted by log probability upon prediction; 12 hyperparameters). Note that *1rest* is arguably the most efficient method which can be used for multi-class, because its binary classifiers can be fitted separately and in parallel. Even if run sequentially, *1rest* requires less memory by a factor of $C$ than a joint multi-class method.

We use our 5-fold CV criterion $\Psi$ for each method. Results here are averaged over ten randomly drawn 5-partitions of the training set (the same partitions are used for the different methods). The test error (in percent) of *mc-sep* is $7.81$ vs. $8.01$ for *1rest*. The result for *mc-sep* is state-of-the-art, for example the best SVM technique tested in [5] attained $7.65$, and SVM one-against-rest attained $8.30$ in this study. Note that while *1rest* also may choose 12 independent kernel parameters, it does not make good use of this possibility, as opposed to *mc-sep*. *mc-tied* has test error $8.37$, suggesting that tying kernels leads to significant degradation. ROC curves for the different methods are given in [8], showing that *mc-sep* also profits from estimating the predictive probabilities in a better way.

## 6.2 Hierarchical Classification: Patent Text Classification

We use the WIPO-alpha collection[7] previously studied in [1], where patents (title and claim text) are to be classified w.r.t. the standard taxonomy *IPC*, a tree with 4 levels and 5229 nodes. Sections A, B,..., H. form the first level. As in [1], we concentrate on the 8 subtasks rooted at the sections, ranging from D ($n = 1140, C = 160, P = 187$) to B ($n = 9794, C = 1172, P = 1319$). We use linear kernels (see Section 5.1) with variance parameters $v_c$. All experiments are averaged over three training/test splits, different methods using the same ones. $\Psi$ is used with a different 5-partition per section and split, the same across all methods. Our method outputs a predictive $\boldsymbol{p}_j \in \mathbb{R}^C$ for each test case $\boldsymbol{x}_j$. The standard prediction $y(\boldsymbol{x}_j) = \operatorname{argmax}_c p_{j,c}$ maximizes expected accuracy, classes are ranked as $r_j(c) \leq r_j(c')$ iff $p_{j,c} \geq p_{j,c'}$. The test scores are the same as in [1]: *accuracy* (acc) $m^{-1}\sum_j \mathrm{I}_{\{y(\boldsymbol{x}_j)=y_j\}}$, *precision* (prec) $m^{-1}\sum_j r_j(y_j)^{-1}$, *parent accuracy* (pacc) $m^{-1}\sum_j \mathrm{I}_{\{\operatorname{par}(y(\boldsymbol{x}_j))=\operatorname{par}(y_j)\}}$, $\operatorname{par}(c)$ being the parent of $L(c)$. Let $\Delta(c, c')$ be half the length of the shortest path between leafs $L(c), L(c')$. The *taxo-loss* (taxo) is $m^{-1}\sum_j \Delta(y(\boldsymbol{x}_j), y_j)$. These scores are motivated in [1]. For taxo-loss and parent accuracy, we better choose $y(\boldsymbol{x}_j)$ to minimize expected loss[8], different from the standard prediction.

We compare methods F1, F2, H1, H2 (F: flat; H: hierarchical). F1: all $v_c$ shared (1); H1: $v_c$ shared across each level of the tree (3). F2, H2: $v_c$ shared across each subtree rooted at root's children (A: 15, B: 34, C: 17, D: 7, E: 7, F: 17, G: 12, H: 5). Recall that there are 3 accuracy parameters. For hyperparameter learning: $k_1 = 8, k_2 = 4, k_3 = 15$ (F1, F2); $k_1 = 10, k_2 = 4, k_3 = 25$ (H1, H2)[9].

|   | acc (%) | | | | prec (%) | | | | taxo | | | |
|---|------|------|------|------|------|------|------|------|------|------|------|------|
|   | F1 | H1 | F2 | H2 | F1 | H1 | F2 | H2 | F1 | H1 | F2 | H2 |
| A | 40.6 | **41.9** | 40.5 | **41.9** | 51.6 | **53.4** | 51.4 | **53.4** | 1.27 | **1.19** | 1.29 | **1.19** |
| B | 32.0 | **32.9** | 31.7 | 32.7 | 41.8 | **43.8** | 41.6 | 43.7 | 1.52 | **1.44** | 1.55 | **1.44** |
| C | 33.7 | **34.7** | 34.1 | 34.5 | 45.2 | **46.6** | 45.4 | 46.4 | 1.34 | **1.26** | 1.35 | 1.27 |
| D | 40.0 | 40.6 | 39.7 | **40.8** | 52.4 | 54.1 | 52.2 | **54.3** | 1.19 | **1.11** | 1.18 | **1.11** |
| E | 33.0 | **34.2** | 32.8 | 34.1 | 45.1 | **47.1** | 45.0 | **47.1** | 1.39 | **1.31** | 1.38 | **1.31** |
| F | 31.4 | 32.4 | 31.4 | **32.5** | 42.8 | 44.9 | 42.8 | **45.0** | 1.43 | **1.34** | 1.43 | **1.34** |
| G | 40.1 | **40.7** | 40.2 | **40.7** | 51.2 | **52.5** | 51.3 | **52.5** | 1.32 | **1.26** | 1.32 | **1.26** |
| H | 39.3 | 39.6 | 39.4 | **39.7** | 52.4 | 53.3 | 52.5 | **53.4** | 1.17 | 1.15 | 1.17 | **1.14** |

|   | taxo[0-1] | | | | pacc (%) | | | | pacc[0-1] (%) | | | |
|---|------|------|------|------|------|------|------|------|------|------|------|------|
|   | F1 | H1 | F2 | H2 | F1 | H1 | F2 | H2 | F1 | H1 | F2 | H2 |
| A | 1.28 | 1.19 | 1.29 | **1.18** | 58.9 | **61.6** | 58.2 | 61.5 | 57.2 | 61.3 | 56.9 | **61.4** |
| B | 1.54 | **1.44** | 1.56 | **1.44** | 53.6 | 56.4 | 52.7 | **56.6** | 51.9 | **55.9** | 51.4 | **55.9** |
| C | 1.33 | **1.26** | 1.32 | **1.26** | 58.9 | **62.6** | 58.5 | 62.0 | 58.6 | **61.8** | 58.9 | 61.6 |
| D | 1.20 | **1.12** | 1.22 | **1.12** | 64.6 | 67.0 | 64.4 | **67.1** | 63.5 | **67.1** | 62.6 | 67.0 |
| E | 1.43 | **1.33** | 1.44 | 1.34 | 56.0 | 59.1 | 56.2 | **59.2** | 54.0 | **58.2** | 53.5 | 57.9 |
| F | 1.43 | **1.34** | 1.44 | **1.34** | 56.8 | 59.7 | 56.8 | **59.8** | 54.9 | 58.7 | 54.6 | **58.9** |
| G | 1.32 | **1.26** | 1.32 | **1.26** | 58.0 | **59.7** | 57.6 | 59.6 | 56.8 | **59.2** | 56.6 | 58.9 |
| H | 1.19 | 1.16 | 1.19 | **1.15** | 61.6 | **62.5** | 61.8 | **62.5** | 59.9 | 61.6 | 60.0 | **61.8** |

Table 1: Results on tasks A-H. Methods F1, F2 flat, H1, H2 hierarchical. taxo[0-1], pacc[0-1] for $\mathrm{argmax}_c\, p_{j,c}$ rule, rather than minimize expected loss.

|   | Final NR (s) | | CV Fold (s) | |   | Final NR (s) | | CV Fold (s) | |
|---|------|------|------|------|---|------|------|------|------|
|   | F1 | H1 | F1 | H1 |   | F1 | H1 | F1 | H1 |
| A | 2030 | 3873 | 573 | 598 | E | 131.5 | 203.4 | 32.2 | 49.6 |
| B | 3751 | 8657 | 873 | 1720 | F | 1202 | 2871 | 426 | 568 |
| C | 4237 | 7422 | 719 | 1326 | G | 1342 | 2947 | 232 | 579 |
| D | 56.3 | 118.5 | 9.32 | 20.2 | H | 971.7 | 1052 | 146 | 230 |

Table 2: Running times for tasks A-H. Method F1 flat, H1 hierarchical. CV Fold: Re-optimization of $\boldsymbol{\alpha}_{[J]}$, gradient accumulation for single fold.

For final fitting: $k_1 = 25$, $k_2 = 12$ (F1, F2); $k_1 = 30$, $k_2 = 17$ (H1, H2). The optimization is started from $v_c = 5$ for all methods. Results are given in Table 1.

The hierarchical model outperforms the flat one consistently. While the differences in accuracy and precision are hardly significant (as also found in [1]), they (partly) are in taxo-loss and parent accuracy. Also, minimizing expected loss is consistently better than using the standard rule for the latter, although the differences are very small. H1 and H2 do not perform differently: choosing many different $v_c$ in the linear kernel seems no advantage here (but see Section 6.1). The results are very similar to the ones of [1]. However, for our method, the recommendation in [1] to use $v_c = 1$ leads to significantly worse results in all scores, the $v_c$ chosen by our methods are generally larger.

In Table 2, we present running times[10] for the final fitting and for a single fold during hyperparameter optimization (5 of them are required for $\Psi$, $\nabla_h \Psi$). Cai and Hofmann [1] quote a final fitting time of $2200s$ on the D section, while we require $119s$ (more than 18 times faster). It is precisely this high efficiency of primary fitting which allows us to use it as inner loop for hyperparameter learning.

## 7 Discussion

We presented a general framework for very efficient large scale kernel multi-way classification with structured label spaces and demonstrated its features on hierarchical text classification tasks with many classes. As shown for the hierarchical case, the framework is easily extended to novel struc-

tural priors or covariance functions, and while not shown here, it is also easy to extend it to different likelihoods (as long as they are log-concave). We solve the kernel parameter learning problem by optimizing the CV log likelihood, whose gradient can be computed within the framework. Our method provides estimates of the predictive distribution at test points, which may result in better predictions for non-standard losses or ROC curves. Efficient and easily extendable code is publicly available (see Section 1).

An extension to multi-label classification is planned. More advanced label set structures can be adressed, noting that Hessian vector products can often be computed in about the same way as gradients. An application to label sequence learning is work in progress, which may even be combined with a hierarchical prior. Infering a hierarchy from data is possible in principle, using expectation maximization techniques (note that the primary fitting can deal with target *distributions* $y_i$), as well as incorporating uncertain data.

Empirical Bayesian methods or approximate CV scores for hyperparameter learning have been proposed in [11, 3, 6], but they are orders of magnitude more expensive than our proposal here, and do not apply to a massive number of classes. Many multi-class SVM techniques are available (see [2, 5] for references). Here, fitting is a constrained convex problem, and often fairly sparse solutions (many zeros in $\alpha$) are found. However, if the degree of sparsity is not large, first-order conditional gradient methods typically applied can be slow[11]. SVM methods typically do not come with efficient automatic kernel parameter learning schemes, and they do not provide estimates of predictive probabilities which are asymptotically correct.

**Acknowledgments**

Thanks to Olivier Chapelle for many useful discussions. Supported in part by the IST Programme of the European Community, under the PASCAL Network of Excellence, IST-2002-506778.

## Footnotes

[1]See www.kyb.tuebingen.mpg.de/bs/people/seeger/lhotse/.

[2]In Matlab, reshape(y,n,C) would give the matrix $(y_{i,c}) \in \mathbb{R}^{n,C}$.

[3]Learning an unknown label set structure may be achieved by expectation maximization techniques, but this is subject to future work.

[4]Initial experiments with conjugate gradients in $\boldsymbol{\alpha}$ gave very slow convergence, due to poor conditioning, but experiments with a different dual criterion are in preparation.

[5]The innermost vector operations work on contiguous chunks of memory, rather than strided ones, thus supporting cacheing or vector functions of the processor.

[6] Available at http://www.niaad.liacc.up.pt/old/statlog/.

[7] Raw data from www.wipo.int/ibis/datasets. Label hierarchy described at www.wipo.int/classifications/en. Thanks to L. Cai, T. Hofmann for providing us with the count data and dictionary. We did Porter stemming, stop word removal, and removal of empty categories. The attributes are bag-of-words over the dictionary of occuring words. All cases $\boldsymbol{x}_i$ were scaled to unit norm.

[8] For parent accuracy, let $p(j)$ be the node with maximal mass (under $\boldsymbol{p}_j$) of its children which are leafs, then $y(\boldsymbol{x}_j)$ must be a child of $p(j)$.

[9] Except for section C, where $k_1 = 14, k_2 = 6, k_3 = 35$.

[10]Processor time on 64bit 2.33GHz AMD machines.

[11]These methods solve a very large number of small problems iteratively, as opposed to ours which does few expensive Newton steps. The latter kind, if feasible at all, often makes better use of hardware features such as cacheing and vector operations, and therefore is the preferred approach in numerical optimization.

# References

[1] L. Cai and T. Hofmann. Hierarchical document categorization with support vector machines. In *CIKM 13*, pages 78–87, 2004.

[2] K. Crammer and Y. Singer. On the algorithmic implementation of multiclass kernel-based vector machines. *J. M. Learn. Res.*, 2:265–292, 2001.

[3] P. Craven and G. Wahba. Smoothing noisy data with spline functions: Estimating the correct degree of smoothing by the method of generalized cross-validation. *Numerische Mathematik*, 31:377–403, 1979.

[4] P.J. Green and B. Silverman. *Nonparametric Regression and Generalized Linear Models*. Monographs on Statistics and Probability. Chapman & Hall, 1994.

[5] C.-W. Hsu and C.-J. Lin. A comparison of methods for multi-class support vector machines. *IEEE Transactions on Neural Networks*, 13:415–425, 2002.

[6] Y. Qi, T. Minka, R. Picard, and Z. Ghahramani. Predictive automatic relevance determination by expectation propagation. In *Proceedings of ICML 21*, 2004.

[7] M. Seeger. Gaussian processes for machine learning. *International Journal of Neural Systems*, 14(2):69–106, 2004.

[8] M. Seeger. Cross-validation optimization for structured Hessian kernel methods. Technical report, Max Planck Institute for Biologic Cybernetics, Tübingen, Germany, 2006. See www.kyb.tuebingen.mpg.de/bs/people/seeger.

[9] Y. Shen, A. Ng, and M. Seeger. Fast Gaussian process regression using KD-trees. In *Advances in NIPS 18*, 2006.

[10] A. Smola and P. Bartlett. Sparse greedy Gaussian process regression. In *Advances in NIPS 13*, pages 619–625, 2001.

[11] C. K. I. Williams and D. Barber. Bayesian classification with Gaussian processes. *IEEE PAMI*, 20(12):1342–1351, 1998.

[12] C. Yang, R. Duraiswami, and L. Davis. Efficient kernel machines using the improved fast Gauss transform. In *Advances in NIPS 17*, pages 1561–1568, 2005.

